# Where are they looking?

**Adrià Recasens**\*    **Aditya Khosla**\*    **Carl Vondrick**    **Antonio Torralba**
Massachusetts Institute of Technology
{recasens, khosla, vondrick, torralba}@csail.mit.edu
(\* - indicates equal contribution)

## Abstract

Humans have the remarkable ability to follow the gaze of other people to identify what they are looking at. Following eye gaze, or *gaze-following*, is an important ability that allows us to understand what other people are thinking, the actions they are performing, and even predict what they might do next. Despite the importance of this topic, this problem has only been studied in limited scenarios within the computer vision community. In this paper, we propose a deep neural network-based approach for gaze-following and a new benchmark dataset, GazeFollow, for thorough evaluation. Given an image and the location of a head, our approach follows the gaze of the person and identifies the object being looked at. Our deep network is able to discover how to extract head pose and gaze orientation, and to select objects in the scene that are in the predicted line of sight and likely to be looked at (such as televisions, balls and food). The quantitative evaluation shows that our approach produces reliable results, even when viewing only the back of the head. While our method outperforms several baseline approaches, we are still far from reaching human performance on this task. Overall, we believe that gaze-following is a challenging and important problem that deserves more attention from the community.

## 1   Introduction

You step out of your house and notice a group of people looking up. You look up and realize they are looking at an aeroplane in the sky. Despite the object being far away, humans have the remarkable ability to precisely follow the gaze direction of another person, a task commonly referred to as *gaze-following* (see [3] for a review). Such an ability is a key element to understanding what people are doing in a scene and their intentions. Similarly, it is crucial for a computer vision system to have this ability to better understand and interpret people. For instance, a person might be holding a book but looking at the television, or a group of people might be looking at the same object which can indicate that they are collaborating at some task, or they might be looking at different places which can indicate that they are not familiar with each other or that they are performing unrelated tasks

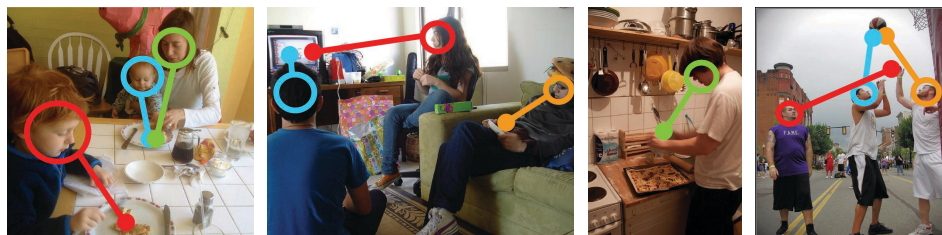

Figure 1: **Gaze-following:** We present a model that learns to predict where people in images are looking. We also introduce GazeFollow, a new large-scale annotated dataset for gaze-following.

(see Figure 1). Gaze-following has applications in robotics and human interaction interfaces where it is important to understand the object of interest of a person. Gaze-following can also be used to predict what a person will do next as people tend to attend to objects they are planning to interact with even before they start an action.

Despite the importance of this topic, only a few works in computer vision have explored gaze-following [5, 16, 14, 15, 18]. Previous work on gaze-following addresses the problem by limiting the scope (e.g., people looking at each other only [14]), by restricting the situations (e.g., scenes with multiple people only or synthetic scenarios [9, 7]), or by using complex inputs (multiple images [5, 15, 18] or eye-tracking data [6]). Only [16] tackles the unrestricted gaze-following scenario but relies on face detectors (therefore can not handle situations such as people looking away from the camera) and is not evaluated on a gaze-following task. Our goal is to perform gaze-following in natural settings without making restrictive assumptions and when only a single view is available. We want to address the general gaze-following problem to be able to handle situations in which several people are looking at each other, and one or more people are interacting with one or more objects.

In this paper, we formulate the problem of gaze-following as: given a single picture containing one or more people, the task is to the predict the location that each person in a scene is looking at. To address this problem, we introduce a deep architecture that learns to combine information about the head orientation and head location with the scene content in order to follow the gaze of a person inside the picture. The input to our model is a picture and the location of the person for who we want to follow the gaze, and the output is a distribution over possible locations that the selected person might be looking at. This output distribution can be seen as a saliency map from the point of view of the person inside the picture. To train and evaluate our model, we also introduce GazeFollow, a large-scale benchmark dataset for gaze-following. Our model, code and dataset are available for download at http://gazefollow.csail.mit.edu.

**Related Work (Saliency):** Although strongly related, there are a number of important distinctions between gaze-following [3] and saliency models of attention [8]. In traditional models of visual attention, the goal is to predict the eye fixations of an observer *looking at a picture*, while in gaze-following the goal is to estimate what is being looked at by a person *inside a picture*. Most saliency models focus on predicting fixations while an observer is free-viewing an image [8, 11] (see [2] for a review). However, in gaze-following, the people in the picture are generally engaged in a task or navigating an environment and, therefore, are not free-viewing and might fixate on objects even when they are not the most salient. A model for gaze-following has to be able to follow the line of sight and then select, among all possible elements that cross the line of sight, which objects are likely to be the center of attention. Both tasks (gaze-following and saliency modeling) are related in several interesting ways. For instance, [1] showed that gaze-following of people inside a picture can influence the fixations of an observer looking at the picture as the object being fixated by the people inside the picture will attract the attention of the observer of the picture.

**Related Work (Gaze):** The work on gaze-following in computer vision is very limited. Gaze-following is used in [16] to improve models of free-viewing saliency prediction. However, they only estimate the gaze direction without identifying the object being attended. Further, their reliance on a face detector [23] prevents them from being able to estimate gaze for people looking away from the camera. Another way of approaching gaze-following is using a wearable eye-tracker to precisely measure the gaze of several people in a scene. For instance, [6] used an eye tracker to predict the next object the user will interact with, and to improve action recognition in egocentric vision. In [14] they propose detecting people looking at each other in a movie in order to better identify interactions between people. As in [16], this work only relies on the direction of gaze without estimating the object being attended, and, therefore, cannot address the general problem of gaze-following, in which a person is interacting with an object. In [5], they perform gaze-following in scenes with multiple observers in an image by finding the regions in which multiple lines of sight intersect. Their method needs multiple people in the scene, each with an egocentric camera, used to get 3D head location, as the model only uses head orientation information and does not incorporate knowledge about the content of the scene. In [15, 18], the authors propose a system to infer the region attracting the attention of a group of people (social saliency prediction). As in [5] their method takes as input a set of pictures taken from the viewpoint of each of the people present in the image and it does not perform gaze-following. Our method only uses a single third-person view of the scene to infer gaze.

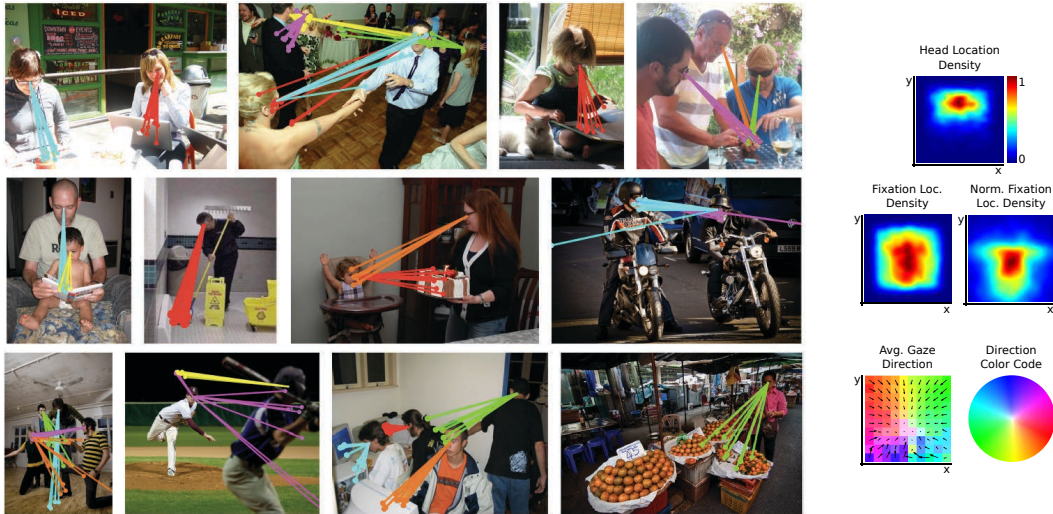

| (a) Example test images and annotations | (b) Test set statistics |

Figure 2: **GazeFollow Dataset:** We introduce a new dataset for gaze-following in natural images. On the left, we show several example annotations and images. In the graphs on the right, we summarize a few statistics about test partition of the dataset. The top three heat maps show the probability density for the location of the head, the fixation location, and the fixation location normalized with respect to the head position. The bottom shows the average gaze direction for various head positions.

## 2   GazeFollow: A Large-Scale Gaze-Following Dataset

In order to both train and evaluate models, we built GazeFollow, a large-scale dataset annotated with the location of where people in images are looking. We used several major datasets that contain people as a source of images: $1,548$ images from SUN [19], $33,790$ images from MS COCO [13], $9,135$ images from Actions 40 [20], $7,791$ images from PASCAL [4], $508$ images from the ImageNet detection challenge [17] and $198,097$ images from the Places dataset [22]. This concatenation results in a challenging and large image collection of people performing diverse activities in many everyday scenarios.

Since the source datasets do not have gaze ground-truth, we annotated it using Amazon's Mechanical Turk (AMT). Workers used our online tool to mark the center of a person's eyes and where the worker believed the person was looking. Workers could indicate if the person was looking outside the image or if the person's head was not visible. To control quality, we included images with known ground-truth, and we used these to detect and discard poor annotations. Finally, we obtained $130,339$ people in $122,143$ images, with gaze locations inside the image.

We use about $4,782$ people of our dataset for testing and the rest for training. We ensured that every person in an image is part of the same split, and to avoid bias, we picked images for testing such that the fixation locations were uniformly distributed across the image. Further, to evaluate human consistency on gaze-following, we collected 10 gaze annotations per person for the test set.

We show some example annotations and statistics of the dataset in Fig.2. We designed our dataset to capture various fixation scenarios. For example, some images contain several people with joint attention while others contain people looking at each other. The number of people in the image can vary, ranging from a single person to a crowd of people. Moreover, we observed that while some people have consistent fixation locations others have bimodal or largely inconsistent distributions, suggesting that solutions to the gaze-following problem could be multimodal.

## 3   Learning to Follow Gaze

At a high level, our model is inspired by how humans tend to follow gaze. When people infer where another person is looking, they often first look at the person's head and eyes to estimate their field of view, and subsequently reason about salient objects in their perspective to predict where they are looking. In this section, we present a model that emulates this approach.

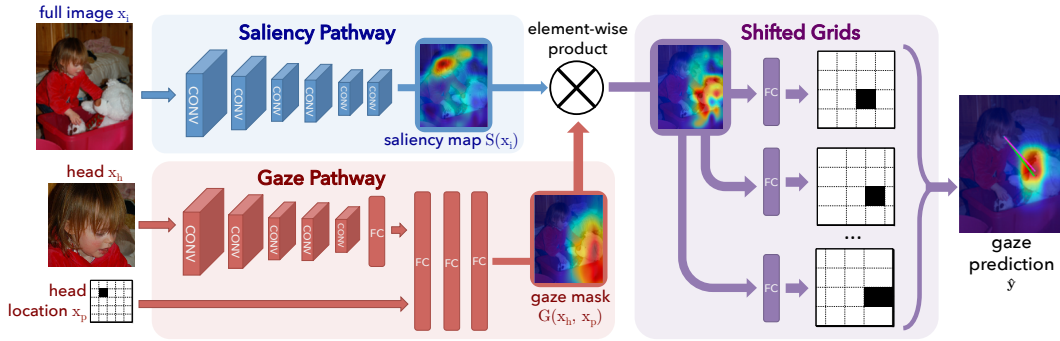

Figure 3: **Network architecture:** We show the architecture of our deep network for gaze-following. Our network has two main components: the saliency pathway (top) to estimate saliency and the gaze pathway (bottom) to estimate gaze direction. See Section 3 for details.

## 3.1 Gaze and Saliency Pathways

Suppose we have an image $x_i$ and a person for whom we want to predict gaze. We parameterize this person with a quantized spatial location of the person's head $x_p$ and a cropped, close-up image of their head $x_h$. Given $x$, we seek to predict the spatial location of the person's fixation $y$. Encouraged by progress in deep learning, we also use deep networks to predict a person's fixation.

Keeping the motivation from Section 3 in mind, we design our network to have two separate pathways for gaze and saliency. The gaze pathway only has access to the closeup image of the person's head and their location, and produces a spatial map, $G(x_h, x_p)$, of size $D \times D$. The saliency pathway sees the full image but not the person's location, and produces another spatial map, $S(x_i)$, of the same size $D \times D$. We then combine the pathways with an element-wise product:

$$\hat{y} = F\left(G(x_h, x_p) \otimes S(x_i)\right)$$

where $\otimes$ represents the element-wise product. $F(\cdot)$ is a fully connected layer that uses the multiplied pathways to predict where the person is looking, $\hat{y}$.

Since the two network pathways only receive a subset of the inputs, they cannot themselves solve the full problem during training, and instead are forced to solve subproblems. Our intention is that, since the gaze pathway only has access to the person's head, $x_h$ and location, $x_p$, we expect it will learn to predict the direction of gaze. Likewise, since the saliency pathway does not know which person to follow, we hope it learns to find objects that are salient, independent of the person's viewpoint. The element-wise product allows these two pathways to interact in a way that is similar to how humans approach this task. In order for a location in the element-wise product to be activated, both the gaze and saliency pathways must have large activations.

**Saliency map:** To form the saliency pathway, we use a convolutional network on the full image to produce a hidden representation of size $D \times D \times K$. Since [21] shows that objects tend to emerge in these deep representations, we can create a gaze-following saliency map by learning the importance of these objects. To do this, we add a convolutional layer that convolves the hidden representation with a $w \in \mathbb{R}^{1 \times 1 \times K}$ filter, which produces the $D \times D$ saliency map. Here, the sign and magnitude of $w$ can be interpreted as weights indicating an object's importance for gaze-following saliency.

**Gaze mask:** In the gaze pathway, we use a convolutional network on the head image. We concatenate its output with the head position and use several fully connected layers and a final sigmoid to predict the $D \times D$ gaze mask.

**Pathway visualization:** Fig. 4 shows examples of the (a) gaze masks and (b) saliency maps learned by our network. Fig. 4(b) also compares the saliency maps of our network with the saliency computed using a state of the art saliency model [11]. Note that our model learns a notion of saliency that is relevant for the gaze-following task and places emphasis on certain objects that people tend to look at (e.g., balls and televisions). In the third example, the red light coming from the computer mouse is salient in the Judd et al [11] model but that object is not relevant in a gaze-following task as the computer monitor is more likely to be the target of attention of the person inside the picture.

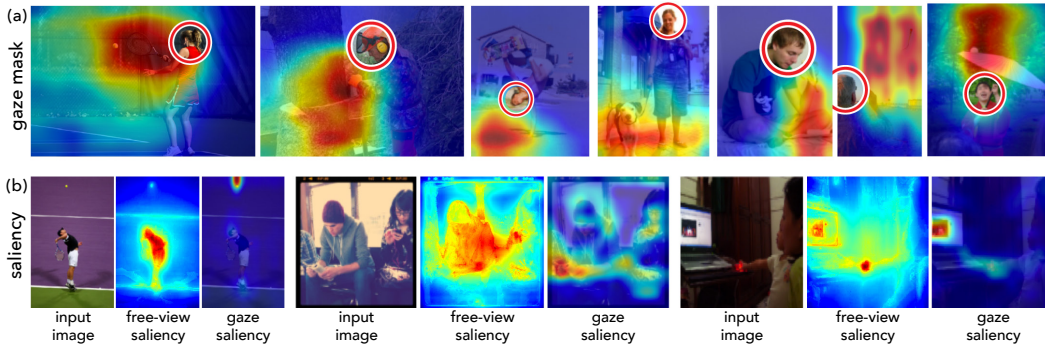

Figure 4: **Pathway visualization:** (a) The gaze mask output by our network for various head poses. (b) Each triplet of images show, from left to right, the input image, its free-viewing saliency estimated using [11], and the gaze-following saliency estimated using our network. These examples clearly illustrate the differences between free-viewing saliency [11] and gaze-following saliency.

## 3.2 Multimodal Predictions

Although humans can often follow gaze reliably, predicting gaze is sometimes ambiguous. If there are several salient objects in the image, or the eye pose cannot be accurately perceived, then humans may disagree when predicting gaze. We can observe this for several examples in Fig. 2. Consequently, we want to design our model to support multimodal predictions.

We could formulate our problem as a regression task (i.e., regress the Cartesian coordinates of fixations) but then our predictions would be unimodal. Instead, we can formulate our problem as a classification task, which naturally supports multimodal outputs because each category has a confidence value. To do this, we quantize the fixation location $y$ into a $N \times N$ grid. Then, the job of the network is to classify the inputs $x$ into one of $N^2$ classes. The model output $\hat{y} \in \mathbb{R}^{N \times N}$ is the confidence that the person is fixating in each grid cell.

**Shifted grids:** For classification, we must choose the number of grid cells, $N$. If we pick a small $N$, our predictions will suffer from poor precision. If we pick a large $N$, there will be more precision, but the learning problem becomes harder because standard classification losses do not gradually penalize spatial categories − a misclassification that is off by just one cell should be penalized less than errors multiple cells away. To alleviate this trade-off, we propose the use of *shifted grids*, as illustrated in Fig. 3, where the network solves several overlapping classification problems. The network predicts locations in multiple grids where each grid is shifted such that cells in one grid overlap with cells in other grids. We then average the shifted outputs to produce the final prediction.

## 3.3 Training

We train our network end-to-end using backpropagation. We use a softmax loss for each shifted grid and average their losses. Since we only supervise the network with gaze fixations, we do not enforce that the gaze and saliency pathways solve their respective subproblems. Rather, we expect that the proposed network structure encourages these roles to emerge automatically (which they do, as shown in Fig. 6).

**Implementation details:** We implemented the network using Caffe [10]. The convolutional layers in both the gaze and saliency pathways follow the architecture of the first five layers of the AlexNet architecture [12]. In our experiments, we initialize these convolutional layers of the saliency pathway with the Places-CNN [22] and those of the gaze pathway with ImageNet-CNN [12]. The last convolutional layer of the saliency pathway has a $1 \times 1 \times 256$ convolution kernel (i.e., $K = 256$). The remaining fully connected layers in the gaze pathway are of sizes 100, 400, 200, and 169 respectively. The saliency map and gaze mask are $13 \times 13$ in size (i.e., $D = 13$), and we use 5 shifted grids of size $5 \times 5$ each (i.e., $N = 5$). For learning, we augment our training data with flips and random crops with the fixation locations adjusted accordingly.

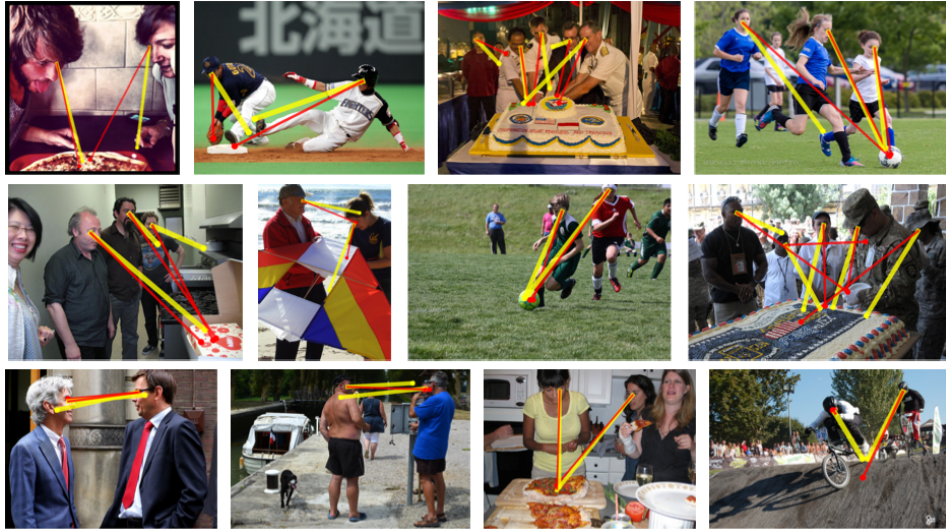

Figure 5: **Qualitative results:** We show several examples of successes and failures of our model. The red lines indicate **ground truth gaze**, and the yellow, our **predicted gaze**.

| Model | AUC | Dist. | Min Dist. | Ang. |
|---|---|---|---|---|
| Our | **0.878** | **0.190** | **0.113** | **24°** |
| SVM+shift grid | 0.788 | 0.268 | 0.186 | 40° |
| SVM+one grid | 0.758 | 0.276 | 0.193 | 43° |
| Judd [11] | 0.711 | 0.337 | 0.250 | 54° |
| Fixed bias | 0.674 | 0.306 | 0.219 | 48° |
| Center | 0.633 | 0.313 | 0.230 | 49° |
| Random | 0.504 | 0.484 | 0.391 | 69° |
| One human | 0.924 | 0.096 | 0.040 | 11° |

(a) Main Evaluation

| Model | AUC | Dist. | Min Dist. | Ang. |
|---|---|---|---|---|
| No image | 0.821 | 0.221 | 0.142 | 27° |
| No position | 0.837 | 0.238 | 0.158 | 32° |
| No head | 0.822 | 0.264 | 0.179 | 41° |
| No eltwise | 0.876 | 0.193 | 0.117 | 25° |
| $5 \times 5$ grid | 0.839 | 0.245 | 0.164 | 36° |
| $10 \times 10$ grid | 0.873 | 0.218 | 0.138 | 30° |
| L2 loss | 0.768 | 0.245 | 0.169 | 34° |
| Our full | 0.878 | 0.190 | 0.113 | 24° |

(b) Model Diagnostics

Table 1: **Evaluation:** (a) We evaluate our model against baselines and (b) analyze how it performances with some components disabled. *AUC* refers to the area under the ROC curve (higher is better). *Dist.* refers to the $L_2$ distance to the average of ground truth fixation, while *Min Dist.* refers to the $L_2$ distance to the nearest ground truth fixation (lower is better). *Ang.* is the angular error of predicted gaze in degrees (lower is better). See Section 4 for details.

# 4 Experiments

## 4.1 Setup

We evaluate the ability of our model to predict where people in images are looking. We use the disjoint train and test sets from GazeFollow, as described in Section 2, to train and evaluate our model. The test set was randomly sampled such that the fixation location was approximately uniform, and ignored people who were looking outside the picture or at the camera. Similar to PASCAL VOC Action Recognition [4] where ground-truth person bounding boxes are available both during training and testing, we assume that we are given the head location at both train and test time. This allows us to focus our attention on the primary task of gaze-following. In Section 4.3, we show that our method performs well even when using a simple head detector.

Our primary evaluation metric compares the ground truth annotations[1] against the distribution predicted by our model. We use the **Area Under Curve (AUC)** criteria from [11] where the predicted heatmap is used as confidences to produce an ROC curve. The AUC is the area under this ROC curve. If our model behaves perfectly, the AUC will be 1 while chance performance is 0.5. $L_2$ **distance:** We evaluate the Euclidean distance between our prediction and the average of ground truth annotations. We assume each image is of size $1 \times 1$ when computing the $L_2$ distance. Additionally, as the ground truth may be multimodal, we also report the minimum $L_2$ distance between our pre-

diction and all ground truth annotations. **Angular error:** Using the ground truth eye position from the annotation we compute the gaze vectors for the average ground truth fixations and our prediction, and report the angular difference between them.

We compare our approach against several baselines ranging from simple (center, fixed bias) to more complex (SVM, free-viewing saliency) as described below. **Center:** The prediction is always the center of the image. **Fixed bias:** The prediction is given by the average of fixations from the training set for heads in similar locations as the test image. **SVM:** We generate features by concatenating the quantized eye position with `pool5` of the ImageNet-CNN [12] for both the full image and the head image. We train a SVM on these features to predict gaze using a similar classification grid setup as our model. We evaluate this approach for both, a single grid and shifted grids. **Free-viewing saliency:** We use a state-of-the-art free-viewing saliency model [11] as a predictor of gaze. Although free-viewing saliency models ignore head orientation and location, they may still identify important objects in the image.

## 4.2 Results

We compare our model against baselines in Tbl.1(a). Our method archives an AUC of $0.878$ and a mean Euclidean error of $0.190$, outperforming all baselines significantly in all the evaluation metrics. The SVM model using shifted grids shows the best baseline performance, surpassing the one grid baseline by a reasonable margin. This verifies the effectiveness of the shifted grids approach proposed in this work.

Fig.5 shows some example outputs of our method. These qualitative results show that our method is able to distinguish people in the image by using the gaze pathway to model a person's point of view, as it produces different outputs for different people in the same image. Furthermore, it is also able to find salient objects in images, such as balls or food. However, the method still has certain limitations. The lack of 3D understanding generates some wrong predictions, as illustrated by the $1^{st}$ image in the $2^{nd}$ row of Fig. 5, where one of the predictions is in a different plane of depth.

To obtain an approximate upper bound on prediction performance, we evaluate human performance on this task. Since we annotated our test set 10 times, we can quantify how well one annotation predicts the mean of the remaining 9 annotations. A single human is able to achieve an AUC of $0.924$ and a mean Euclidean error of $0.096$. While our approach outperforms all baselines, it is still far from reaching human performance. We hope that the availability of GazeFollow will motivate further research in this direction, allowing machines to reach human level performance.

## 4.3 Analysis

**Ablation study:** In Tbl. 1(b), we report the performance after removing different components of our model, one at a time, to better understand their significance. In general, all three of inputs (image, position and head) contribute to the performance of our model. Interestingly, the model with only the head and its position achieves comparable *angular* error to our full method, suggesting that the gaze pathway is largely responsible for estimating the gaze direction. Further, we show the results of our model with single output grids ($5 \times 5$ and $10 \times 10$). Removing shifted grids hurts performance significantly as shifted grids have a spatially graded loss function, which is important for learning.

**Internal representation:** In Fig. 6, we visualize the various stages of our network. We show the output of each of the pathways as well as the element wise product. For example, in the second row we have two different girls writing on the blackboard. The gaze mask effectively creates a heat map of the field of view for the girl in the right, while the saliency map identifies the salient spots in the image. The element-wise multiplication of the saliency map and gaze mask removes the responses of the girl on the left and attenuates the saliency of the right girl's head. Finally, our shifted grids approach accurately predicts where the girl is looking.

Further, we apply the technique from [21] to visualize the top activations for different units in the fifth convolutional layer of the saliency pathway. We use filter weights from the sixth convolutional layer to rank their contribution to the saliency map. Fig. 7 shows four units with positive (left) and negative (right) contributions to the saliency map. Interestingly, $w$ learns positive weights for salient objects such as *switched on TV monitors* and balls, and negative weights for non-salient objects.

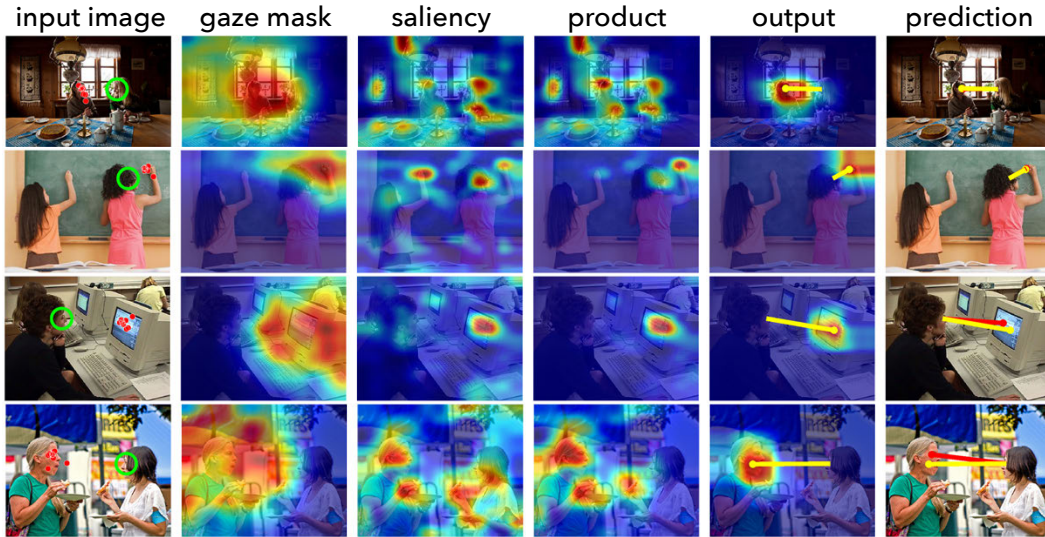

Figure 6: **Visualization of internal representations:** We visualize the output of different components of our model. The green circle indicates the **person whose gaze we are trying to predict**, the red dots/lines show the **ground truth gaze**, and the yellow line is our **predicted gaze**.

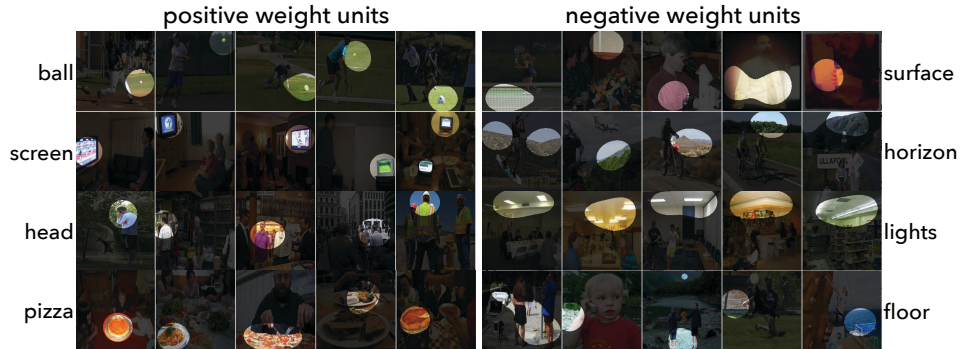

Figure 7: **Visualization of saliency units:** We visualize several units in our saliency pathway by finding images with high scoring activations, similar to [21]. We sort the units by $w$, the weights of the sixth convolutional layer (See Section 3.1 for more details). Positive weights tend to correspond to salient everyday objects, while negative weights tend to correspond to background objects.

**Automatic head detection:** To evaluate the impact of imperfect head locations on our system, we built a simple head detector, and input its detections into our model. For detections surpassing the intersection over union threshold of $0.5$, our model achieved an AUC of $0.868$, as compared to an AUC of $0.878$ when using ground-truth head locations. This demonstrates that our model is robust to inaccurate head detections, and can easily be made fully-automatic.

## 5   Conclusion

Accurate gaze-following achieving human-level performance will be an important tool to enable systems that can interpret human behavior and social situations. In this paper, we have introduced a model that learns to do gaze-following using GazeFollow, a large-scale dataset of human annotated gaze. Our model automatically learns to extract the line of sight from heads, without using any supervision on head pose, and to detect salient objects that people are likely to interact with, without requiring object-level annotations during training. We hope that our model and dataset will serve as important resources to facilitate further research in this direction.

**Acknowledgements.** We thank Andrew Owens for helpful discussions. Funding for this research was partially supported by the Obra Social "la Caixa" Fellowship for Post-Graduate Studies to AR and a Google PhD Fellowship to CV.

## Footnotes

[1]Note that, as mentioned in Section 2, we obtain 10 annotations per person in the test set.

# References

[1] A. Borji, D. Parks, and L. Itti. Complementary effects of gaze direction and early saliency in guiding fixations during free viewing. *Journal of vision*, 14(13):3, 2014.

[2] A. Borji, D. N. Sihite, and L. Itti. Salient object detection: A benchmark. In *ECCV*. 2012.

[3] N. Emery. The eyes have it: the neuroethology, function and evolution of social gaze. *Neuroscience & Biobehavioral Reviews*, 2000.

[4] M. Everingham, L. Van Gool, C. K. Williams, J. Winn, and A. Zisserman. The PASCAL Visual Object Classes (VOC) Challenge. *IJCV*, 2010.

[5] A. Fathi, J. K. Hodgins, and J. M. Rehg. Social interactions: A first-person perspective. In *CVPR*, 2012.

[6] A. Fathi, Y. Li, and J. M. Rehg. Learning to recognize daily actions using gaze. In *ECCV*. 2012.

[7] M. W. Hoffman, D. B. Grimes, A. P. Shon, and R. P. Rao. A probabilistic model of gaze imitation and shared attention. *Neural Networks*, 2006.

[8] L. Itti and C. Koch. Computational modelling of visual attention. *Nature Reviews Neuroscience*, 2001.

[9] H. Jasso, J. Triesch, and G. Deák. Using eye direction cues for gaze following–a developmental model. In *ICDL*, 2006.

[10] Y. Jia. Caffe: An open source convolutional architecture for fast feature embedding. *http://caffe.berkeleyvision.org*, 2013.

[11] T. Judd, K. Ehinger, F. Durand, and A. Torralba. Learning to predict where humans look. In *CVPR*, 2009.

[12] A. Krizhevsky, I. Sutskever, and G. E. Hinton. Imagenet classification with deep convolutional neural networks. In *NIPS*, 2012.

[13] T.-Y. Lin, M. Maire, S. Belongie, J. Hays, P. Perona, D. Ramanan, P. Dollár, and C. L. Zitnick. Microsoft coco: Common objects in context. In *ECCV*. 2014.

[14] M. J. Marin-Jimenez, A. Zisserman, M. Eichner, and V. Ferrari. Detecting people looking at each other in videos. *IJCV*, 2014.

[15] H. Park, E. Jain, and Y. Sheikh. Predicting primary gaze behavior using social saliency fields. In *ICCV*, 2013.

[16] D. Parks, A. Borji, and L. Itti. Augmented saliency model using automatic 3d head pose detection and learned gaze following in natural scenes. *Vision Research*, 2014.

[17] O. Russakovsky, J. Deng, H. Su, J. Krause, S. Satheesh, S. Ma, Z. Huang, A. Karpathy, A. Khosla, M. Bernstein, et al. Imagenet large scale visual recognition challenge. *IJCV*, 2015.

[18] H. Soo Park and J. Shi. Social saliency prediction. In *CVPR*, 2015.

[19] J. Xiao, J. Hays, K. A. Ehinger, A. Oliva, and A. Torralba. SUN database: Large-scale scene recognition from abbey to zoo. In *CVPR*, 2010.

[20] B. Yao, X. Jiang, A. Khosla, A. L. Lin, L. Guibas, and L. Fei-Fei. Human action recognition by learning bases of action attributes and parts. In *ICCV*, 2011.

[21] B. Zhou, A. Khosla, A. Lapedriza, A. Oliva, and A. Torralba. Object detectors emerge in deep scene cnns. In *ICLR*, 2015.

[22] B. Zhou, A. Lapedriza, J. Xiao, A. Torralba, and A. Oliva. Learning deep features for scene recognition using places database. In *NIPS*, 2014.

[23] X. Zhu and D. Ramanan. Face detection, pose estimation, and landmark localization in the wild. In *CVPR*, 2012.

